# Control of Selective Visual Attention: Modeling the "Where" Pathway

**Ernst Niebur***
Computation and Neural Systems 139-74
California Institute of Technology

**Christof Koch**
Computation and Neural Systems 139-74
California Institute of Technology

## Abstract

Intermediate and higher vision processes require selection of a subset of the available sensory information before further processing. Usually, this selection is implemented in the form of a spatially circumscribed region of the visual field, the so-called "focus of attention" which scans the visual scene dependent on the input and on the attentional state of the subject. We here present a model for the control of the focus of attention in primates, based on a saliency map. This mechanism is not only expected to model the functionality of biological vision but also to be essential for the understanding of complex scenes in machine vision.

## 1 Introduction: "What" and "Where" In Vision

It is a generally accepted fact that the computations of early vision are massively parallel operations, i.e., applied in parallel to all parts of the visual field. This high degree of parallelism cannot be sustained in intermediate and higher vision because of the astronomical number of different possible combination of features. Therefore, it becomes necessary to select only a part of the instantaneous sensory input for more detailed processing and to discard the rest. This is the mechanism of visual selective attention.

* Present address: Zanvyl Krieger Mind/Brain Institute and Department of Neuroscience, 3400 N. Charles Street, The Johns Hopkins University, Baltimore, MD 21218.

It is clear that similar selection mechanisms are also required in machine vision for the analysis of all but the simplest visual scenes. Attentional mechanisms are slowly introduced in this field; e.g., Yamada and Cottrell (1995) used sequential scanning by a "focus of attention" in the context of face recognition. Another model for eye scan path generation, which is characterized by a strong top-down influence, is presented by Rao and Ballard (this volume). Sequential scanning can be applied to more abstract spaces, like the dynamics of complex systems in optimization problems with large numbers of minima (Tsioutsias and Mjolsness, this volume).

Primate vision is organized along two major anatomical pathways. One of them is concerned mainly with object *recognition*. For this reason, it has been called the *What*–pathway; for anatomical reasons, it is also known as the ventral pathway. The principal task of the other major pathway is the determination of the *location* of objects and therefore it is called the *Where* pathway or, again for anatomical reasons, the dorsal pathway.

In previous work (Niebur & Koch, 1994), we presented a model for the implementation of the *What* pathway. The underlying mechanism is "temporal tagging:" it is assumed that the attended region of the visual field is distinguished from the unattended parts by the temporal fine-structure of the neuronal spike trains. We have shown that temporal tagging can be achieved by introducing moderate levels of correlation[1] between those neurons which respond to attended stimuli.

How can such synchronicity be obtained? We have suggested a simple, neurally plausible mechanism, namely common input to all cells which respond to attended stimuli. Such (excitatory) input will increase the propensity of postsynaptic cells to fire for a short time after receiving this input, and thereby increase the correlation between spike trains without necessarily increasing the average firing rate.

The subject of the present study is to provide a model of the control system which generates such modulating input. We will show that it is possible to construct an integrated system of attentional control which is based on neurally plausible elements and which is compatible with the anatomy and physiology of the primate visual system. The system scans a visual scene and identifies its most salient parts. A possible task would be "Find all faces in this image." We are confident that this model will not only further our understanding of the function of biological vision but that it will also be relevant for technical applications.

## 2    A Simple Model of The Dorsal Pathway

### 2.1    Overall Structure

Figure 1 shows an overview of the model *Where* pathway. Input is provided in the form of digitized images from an NTSC camera which is then analyzed in various feature maps. These maps are organized around the known operations in early visual cortices. They are implemented at different spatial scales and in a center-surround structure akin to visual receptive fields. Different spatial scales are implemented as Gaussian pyramids (Adelson, Anderson, Bergen, Burt, & Ogden, 1984). The center

of the receptive field corresponds to the value of a pixel at level $n$ in the pyramid and the surround to the corresponding pixels at level $n+2$, level 0 being the image in normal size. The features implemented so far are the three principal components of primate color vision (intensity, red-green, blue-yellow), four orientations, and temporal change. Short descriptions of the different feature maps are presented in the next section (2.2).

We then (section 2.3) address the question of the integration of the input in the "saliency map," a topographically organized map which codes for the instantaneous conspicuity of the different parts of the visual field.

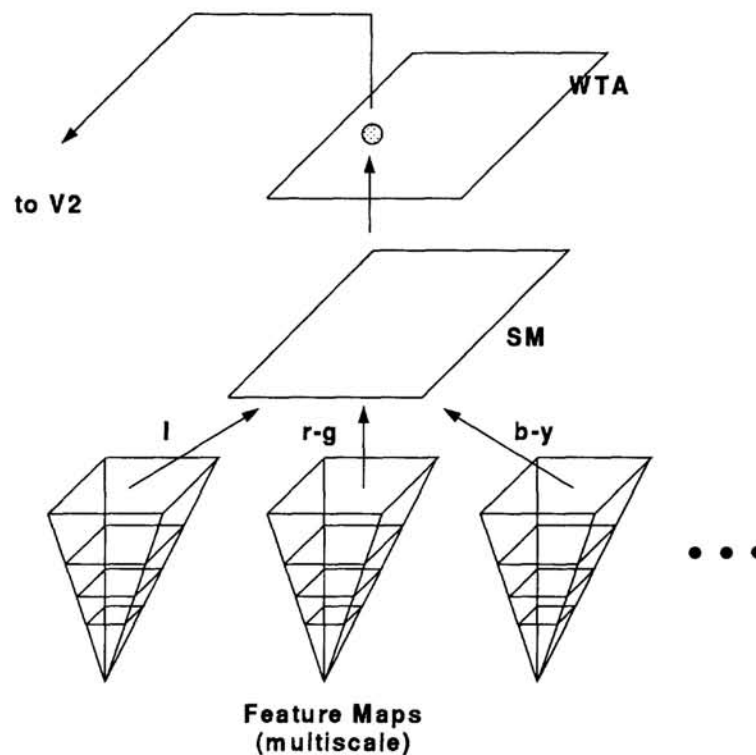

Figure 1: Overview of the model *Where* pathway. Features are computed as center-surround differences at 4 different spatial scales (only 3 feature maps shown). They are combined and integrated in the saliency map ("SM") which provides input to an array of integrate-and-fire neurons with global inhibition. This array ("WTA") has the functionality of a winner-take-all network and provides the output to the ventral pathway ("V2") as well as feedback to the saliency map (curved arrow).

## 2.2   Input Features

### 2.2.1   Intensity

Intensity information is obtained from the chromatic information of the NTSC signal. With $R, G$, and $B$ being the red, green and blue channels, respectively, the intensity $I$ is obtained as $I = (R + G + B)/3$. The entry in the feature map is the modulus

of the contrast, i.e., $|I_{center} - I_{surround}|$. This corresponds roughly to the sum of two single-opponent cells of opposite phase, i.e. bright-center — dark-surround and *vice-versa*. Note, however, that the present version of the model does not reproduce the temporal behavior of ON and OFF subfields because we update the activities in the feature maps instantaneously with changing visual input. Therefore, we neglect the temporal filtering properties of the input neurons.

### 2.2.2   Chromatic Input

Red, green and blue are the pixel values of the RGB signal. Yellow is computed as $(R + G)/2$. At each pixel, we compute a quantity corresponding to the double-opponency cells in primary visual cortex. For instance, for the red-green filter, we first compute at each pixel the value of (red-green). From this, we then subtract (green-red) of the surround. Finally, we take the absolute value of the result.

### 2.2.3   Orientation

The intensity image is convolved with four Gabor patches of angles 0,45,90, and 135 degrees, respectively. The result of these four convolutions are four arrays of scalars at every level of the pyramid. The average orientation is then computed as a weighted vector sum. The components in this sum are the four unit vectors $\vec{u}_i, i = 1, ...4$ corresponding to the 4 orientations, each with the weight $w_i$. This weight is given by the result of its convolution of the respective Gabor patch with the image. Let $\vec{c}$ be this vector for the center pixel, then $\vec{c} = \sum_{i=1}^{4} w_i \vec{u}_i$.

The average orientation vector for the surround, $\vec{s}$, is computed analogously. What enters in the SM is the center-surround difference, i.e. the scalar product $\vec{c}(\vec{s} - \vec{c})$. This is a scalar quantity which corresponds to the center-surround difference in orientation at every location, and which also takes into account the relative "strength" of the oriented edges.

### 2.2.4   Change

The appearance of an object and the segregation of an object from its background have been shown to capture attention, even for stimuli which are equiluminant with the background (Hillstrom & Yantis, 1994). We incorporate the attentional capture by visual onsets and motion by adding the temporal derivative of the input image sequence, taking into account chromatic information. More precisely, at each pixel we compute at time $t$ and for a time difference $\Delta t = 200ms$:

$$\frac{1}{3}\{|R(t) - R(t - \Delta t)| + |G(t) - G(t - \Delta t)| + |B(t) - B(t - \Delta t)|\} \qquad (1)$$

### 2.2.5   Top-Down Information

Our model implements essentially bottom-up strategies for the rapid selection of conspicuous parts of the visual field and does not pretend to be a model for higher cognitive functions. Nevertheless, it is straightforward to incorporate some top-down influence. For instance, in a "Posner task" (Posner, 1980), the subject is instructed to attend selectively to one part of the visual field. This instruction can be implemented by additional input to the corresponding part of the saliency map.

## 2.3  The Saliency Map

The existence of a saliency map has been suggested by Koch and Ullman (1985); see also the "master map" of Treisman (1988). The idea is that of a topographically organized map which encodes information on *where* salient (conspicuous) objects are located in the visual field, but not *what* these objects are.

The task of the saliency map is the computation of the salience at every location in the visual field and the subsequent selection of the most salient areas or objects. At any time, only one such area is selected. The feature maps provide current input to the saliency map. The output of the saliency map consists of a spike train from neurons corresponding to this selected area in the topographic map which project to the ventral ("What") pathway. By this mechanism, they are "tagged" by modulating the temporal structure of the neuronal signals corresponding to attended stimuli (Niebur & Koch, 1994).

### 2.3.1  Fusion Of Information

Once all relevant features have been computed in the various feature maps, they have to be combined to yield the salience, i.e. a scalar quantity. In our model, we solve this task by simply adding the activities in the different feature maps, as computed in section 2.2, with constant weights. We choose all weights identical except for the input obtained from the temporal change. Because of the obvious great importance changing stimuli have for the capture of attention, we select this weight five times larger than the others.

### 2.3.2  Internal Dynamics And Trajectory Generation

By definition, the activity in a given location of the saliency map represents the relative conspicuity of the corresponding location in the visual field. At any given time, the maximum of this map is therefore the most salient stimulus. As a consequence, this is the stimulus to which the focus of attention should be directed next to allow more detailed inspection by the more powerful "higher" process which are not available to the massively parallel feature maps. This means that we have to determine the instantaneous maximum of this map.

This maximum is selected by application of a winner-take-all mechanism. Different mechanisms have been suggested for the implementation of neural winner-take-all networks (e.g., Koch & Ullman, 1985; Yuille & Grzywacz, 1989). In our model, we used a 2-dimensional layer of integrate-and-fire neurons with strong global inhibition in which the inhibitory population is reliably activated by any neuron in the layer. Therefore, when the first of these cells fires, it will inhibit all cells (including itself), and the neuron with the strongest input will generate a sequence of action potentials. All other neurons are quiescent.

For a static image, the system would so far attend continuously the most conspicuous stimulus. This is neither observed in biological vision nor desirable from a functional point of view; instead, after inspection of any point, there is usually no reason to dwell on it any longer and the next-most salient point should be attended.

We achieve this behavior by introducing feedback from the winner-take-all array. When a spike occurs in the WTA network, the integrators in the saliency map

receive additional input with the spatial structure of an inverted Mexican hat, ie. a difference of Gaussians. The (inhibitory) center is at the location of the winner which becomes thus inhibited in the saliency map and, consequently, attention switches to the next-most conspicuous location. The function of the positive lobes of the inverted Mexican hat is to avoid excessive jumping of the focus of attention. If two locations are of nearly equal conspicuity and one of them is close to the present focus of attention, the next jump will go to the close location rather than to the distant one.

## 3   Results

We have studied the system with inputs constructed analogously to typical visual psychophysical stimuli and obtained results in agreement with experimental data. Space limitations prevent a detailed presentation of these results in this report. Therefore, in Fig. 2, we only show one example of a "real-world image." We choose, as an example, an image showing the Caltech bookstore and the trajectory of the focus of attention follows in our model. The most salient feature in this image is the red banner on the the wall of the building (in the center of the image). The focus of attention is directed first to this salient feature. The system then starts to scan the image in the order of decreasing saliency. Shown are the 3 jumps following the initial focussing on the red banner. The jumps are driven by a strong inhibition-of-return mechanism. Experimental evidence for such a mechanims has been obtained recently in area 7a of rhesus monkeys (Steinmetz, Connor, Constantinidis, & McLaughlin, 1994).

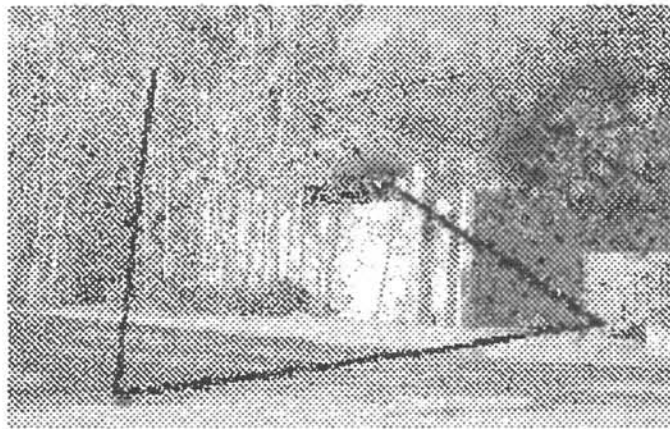

Figure 2: Example image. The black line shows the trajectory of the simulated focus of attention over a time of 140 ms which jumps from the center (red banner on wall of building) to three different locations of decreasing saliency.

## 4   Conclusion And Outlook

We present in this report a prototype for an integrated system mimicking the control of visual selective attention. Our model is compatible with the known anatomy and physiology of the primate visual system, and its different parts communicate by signals which are neurally plausible. The model identifies the most salient points in a visual scenes one-by-one and scans the scene autonomously in the order of

decreasing saliency. This allows the control of a subsequently activated processor which is specialized for detailed object recognition.

At present, saliency is determined by combining the input from a set of feature maps with fixed weights. In future work, we will generalize our approach by introducing plasticity in these weights and thus adapting the system to the task at hand.

### Acknowledgements

Work supported by the Office of Naval Research, the Air Force Office of Scientific Research, the National Science Foundation, the Center for Neuromorphic Systems Engineering as a part of the National Science Foundation Engineering Research Center Program, and by the Office of Strategic Technology of the California Trade and Commerce Agency.

## Footnotes

[1]In (Niebur, Koch, & Rosin, 1993), a similar model was developed using periodic "40Hz" modulation. The present model can be adapted *mutatis mutandis* to this type of modulation.

# References

Adelson, E., Anderson, C., Bergen, J., Burt, P., & Ogden, J. (1984). Pyramid methods in image processing. *RCA Engineer, Nov-Dec.*

Hillstrom, A. & Yantis, S. (1994). Visual motion and attentional capture. *Perception & Psychophysics, 55*(4), 399–411.

Koch, C. & Ullman, S. (1985). Shifts in selective visual attention: towards the underlying neural circuitry. *Human Neurobiol., 4,* 219–227.

Niebur, E. & Koch, C. (1994). A model for the neuronal implementation of selective visual attention based on temporal correlation among neurons. *Journal of Computational Neuroscience, 1*(1), 141–158.

Niebur, E., Koch, C., & Rosin, C. (1993). An oscillation-based model for the neural basis of attention. *Vision Research, 33,* 2789–2802.

Posner, M. (1980). Orienting of attention. *Quart. J. Exp. Psychol., 32,* 3–25.

Steinmetz, M., Connor, C., Constantinidis, C., & McLaughlin, J. (1994). Covert attention suppresses neuronal responses in area 7A of the posterior parietal cortex. *J. Neurophysiology, 72,* 1020–1023.

Treisman, A. (1988). Features and objects: the fourteenth Bartlett memorial lecture. *Quart. J. Exp. Psychol., 40A,* 201–237.

Tsioutsias, D. I. & Mjolsness, E. (1996). A Multiscale Attentional Framework for Relaxation Neural Networks. In Touretzky, D., Mozer, M. C., & Hasselmo, M. E. (Eds.), *Advances in Neural Information Processing Systems,* Vol. 8. MIT Press, Cambridge, MA.

Yamada, K. & Cottrell, G. W. (1995). A model of scan paths applied to face recognition. In *Proc. 17th Ann. Cog. Sci. Conf.* Pittsburgh.

Yuille, A. & Grzywacz, N. (1989). A winner-take-all mechanism based on presynaptic inhibition feedback. *Neural Computation, 2,* 334–344.
